# On the Adaptive Properties of Decision Trees

**Clayton Scott**
Statistics Department
Rice University
Houston, TX 77005
cscott@rice.edu

**Robert Nowak**
Electrical and Computer Engineering
University of Wisconsin
Madison, WI 53706
nowak@engr.wisc.edu

## Abstract

Decision trees are surprisingly adaptive in three important respects: They automatically (1) *adapt* to favorable conditions near the Bayes decision boundary; (2) *focus* on data distributed on lower dimensional manifolds; (3) *reject* irrelevant features. In this paper we examine a decision tree based on dyadic splits that adapts to each of these conditions to achieve minimax optimal rates of convergence. The proposed classifier is the first known to achieve these optimal rates while being practical and implementable.

## 1 Introduction

This paper presents three adaptivity properties of decision trees that lead to faster rates of convergence for a broad range of pattern classification problems. These properties are:

**Noise Adaptivity:** Decision trees can automatically adapt to the (unknown) regularity of the excess risk function in the neighborhood of the Bayes decision boundary. The regularity is quantified by a condition similar to Tsybakov's noise condition [1].

**Manifold Focus:** When the distribution of features happens to have support on a lower dimensional manifold, decision trees can automatically detect and adapt their structure to the manifold. Thus decision trees learn the "effective" data dimension.

**Feature Rejection:** If certain features are irrelevant (i.e., independent of the class labels), then decision trees can automatically ignore these features. Thus decision trees learn the "relevant" data dimension.

Each of the above properties can be formalized and translated into a class of distributions with known minimax rates of convergence. Adaptivity is a highly desirable quality of classifiers since in practice the precise characteristics of the distribution are unknown.

We show that *dyadic decision trees* achieve the (minimax) optimal rate (to within a log factor) without needing to know the specific parameters defining the class. Such trees are constructed by minimizing a complexity penalized empirical risk over an appropriate family of dyadic partitions. The complexity term is derived from a new generalization error bound for trees, inspired by [2]. This bound in turn leads to an oracle inequality from which the optimal rates are derived. Full proofs of all results are given in [11].

The restriction to dyadic splits is necessary to achieve a computationally tractable classifier. Our classifiers have computational complexity nearly linear in the training sample size. The same rates may be achieved by more general tree classifiers, but these require searches over prohibitively large families of partitions. Dyadic decision trees are thus preferred because they are simultaneously implementable, analyzable, and sufficiently flexible to achieve optimal rates.

## 1.1 Notation

Let $Z$ be a random variable taking values in a set $\mathcal{Z}$, and let $Z^n = \{Z_1, \ldots, Z_n\}$ be iid realizations of $Z$. Let $\mathbf{P}_Z$ be the probability measure for $Z$, and let $\widehat{\mathbf{P}}_n$ be the empirical estimate of $\mathbf{P}_Z$ based on $Z^n$: $\widehat{\mathbf{P}}_n(B) = (1/n)\sum_{i=1}^n I_{\{Z_i \in B\}}$, $B \subseteq \mathcal{Z}$, where $I$ denotes the indicator function. In classification we take $\mathcal{Z} = \mathcal{X} \times \mathcal{Y}$, where $\mathcal{X}$ is the collection of feature vectors and $\mathcal{Y}$ is a finite set of class labels. Assume $\mathcal{X} = [0,1]^d$, $d \geq 2$, and $\mathcal{Y} = \{0,1\}$. A classifier is a measurable function $f : [0,1]^d \to \{0,1\}$. Each classifier $f$ induces a set $B_f = \{(x,y) \in \mathcal{Z} \mid f(x) \neq y\}$. Define the probability of error and empirical error (risk) of $f$ by $R(f) = \mathbf{P}_Z(B_f)$ and $\widehat{R}_n(f) = \widehat{\mathbf{P}}_n(B_f)$, respectively. The Bayes classifier $f^*$ achieves minimum probability of error and is given by $f^*(x) = I_{\{\eta(x) > 1/2\}}$, where $\eta(x) = \mathbf{P}_{Y|X}(1 \mid x)$ is the posterior probability that the correct label is 1. The Bayes error is $R(f^*)$ and denoted $R^*$. The Bayes decision boundary, denoted $\partial G^*$, is the topological boundary of the Bayes decision set $G^* = \{x \mid f^*(x) = 1\}$.

## 1.2 Rates of Convergence in Classification

In this paper we study the rate at which $\mathbf{E}_{Z^n}\{R(\widehat{f}_n)\} - R^*$ goes to zero as $n \to \infty$, where $\widehat{f}_n$ is a classification learning rule, i.e., a rule for constructing a classifier from $Z^n$. Yang [3] shows that for $\eta(x)$ in certain smoothness classes minimax optimal rates are achieved by appropriate plug-in density estimates. Tsybakov and collaborators replace global restrictions on $\eta$ by restrictions on $\eta$ near $\partial G^*$. Faster rates are then possible, although existing optimal classifiers typically rely on $\epsilon$-nets or otherwise non-implementable methods. [1, 4, 5]. Other authors have derived rates of convergence for existing practical classifiers, but these rates are suboptimal in the minimax sense considered here [6–8]. Our contribution is to demonstrate practical classifiers that adaptively attain minimax optimal rates for some of Tsybakov's and other classes.

## 2 Dyadic Decision Trees

A *dyadic decision tree* (DDT) is a decision tree that divides the input space by means of axis-orthogonal dyadic splits. More precisely, a dyadic decision tree $T$ is specified by assigning an integer $s(v) \in \{1, \ldots, d\}$ to each internal node $v$ of $T$ (corresponding to the coordinate/attribute that is split at that node), and a binary label 0 or 1 to each leaf node.

The nodes of DDTs correspond to hyperrectangles (cells) in $[0,1]^d$ (see Figure 1). Given a hyperrectangle $A = \prod_{r=1}^d [a_r, b_r]$, let $A^{s,1}$ and $A^{s,2}$ denote the hyperrectangles formed by splitting $A$ at its midpoint along coordinate $s$. Specifically, define $A^{s,1} = \{x \in A \mid x^s \leq (a_s + b_s)/2\}$ and $A^{s,2} = A \backslash A^{s,1}$. Each node of a DDT is associated with a cell according to the following rules: (1) The root node is associated with $[0,1]^d$; (2) If $v$ is an internal node associated to the cell $A$, then the children of $v$ are associated to $A^{s(v),1}$ and $A^{s(v),2}$.

Let $\pi(T) = \{A_1, \ldots, A_k\}$ denote the partition induced by $T$. Let $j(A)$ denote the depth of $A$ and note that $\lambda(A) = 2^{-j(A)}$ where $\lambda$ is the Lebesgue measure on $\mathbb{R}^d$. Define $\mathcal{T}$ to be the collection of all DDTs and $\mathcal{A}$ to be the collection of all cells corresponding to nodes

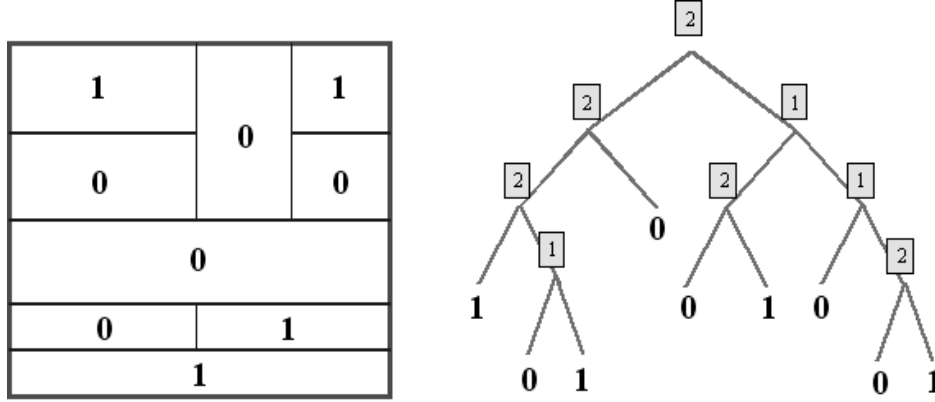

Figure 1: A dyadic decision tree (right) with the associated recursive dyadic partition (left) when $d = 2$. Each internal node of the tree is labeled with an integer from $1$ to $d$ indicating the coordinate being split at that node. The leaf nodes are decorated with class labels.

of trees in $\mathcal{T}$.

Let $M$ be a dyadic integer, that is, $M = 2^L$ for some nonnegative integer $L$. Define $\mathcal{T}_M$ to be the collection of all DDTs such that no terminal cell has a sidelength smaller than $2^{-L}$. In other words, no coordinate is split more than $L$ times when traversing a path from the root to a leaf.

We will consider classifiers of the form

$$\widehat{T}_n = \underset{T \in \mathcal{T}_M}{\arg\min} \; \widehat{R}_n(T) + \Phi_n(T) \tag{1}$$

where $\Phi_n$ is a "penalty" or regularization term specified below. An algorithm of Blanchard et al. [9] may be used to compute $\widehat{T}_n$ in $O(ndL^d \log(ndL^d))$ operations. For all of our theorems on rates of convergence below we have $L = O(\log n)$, in which case the computational cost is $O(nd(\log n)^{d+1})$.

## 3   Generalization Error Bounds for Trees

In this section we state a uniform error bound and an oracle inequality for DDTs. These two results are extensions of our previous work on DDTs [10]. The bounding techniques are quite general and can be extended to larger (even uncountable) families of trees using VC theory, but for the sake of simplicity we confine the discussion to DDTs. Complete proofs may be found in [11]. Before stating these results, some additional notation is necessary.

Let $A \in \mathcal{A}$, and define $[\![A]\!] = (2 + \log_2 d)j(A)$. $[\![A]\!]$ represents the number of bits needed to uniquely encode $A$ and will be used to measure the complexity of a DDT having $A$ as a leaf cell. These "codelengths" satisfy a Kraft inequality $\sum_{A \in \mathcal{A}} 2^{-[\![A]\!]} \leq 1$.

For a cell $A \subseteq [0, 1]^d$, define $p_A = \mathbf{P}_X(A)$ and $\hat{p}_A = (1/n) \sum_{i=1}^n I_{\{X_i \in A\}}$. Further define $\hat{p}'_A = 4 \max(\hat{p}_A, ([\![A]\!] \log 2 + \log n)/n)$ and $p'_A = 4 \max(p_A, ([\![A]\!] \log 2 + \log n)/(2n))$. It can be shown that with high probability, $p_A \leq \hat{p}'_A$ and $\hat{p}_A \leq p'_A$ uniformly over all $A \in \mathcal{A}$ [11]. The mutual boundedness of $p_A$ and $\hat{p}_A$ is a key to making our proposed classifier both computable on the one hand and analyzable on the other.

Define the data-dependent penalty

$$\Phi_n(T) = \sum_{A \in \pi(T)} \sqrt{2\hat{p}'_A \frac{[\![A]\!] \log 2 + \log(2n)}{n}}. \tag{2}$$

Our first main result is the following uniform error bound.

**Theorem 1.** *With probability at least* $1 - 2/n$,

$$R(T) \leq \widehat{R}_n(T) + \Phi_n(T) \quad \textit{for all } T \in \mathcal{T}. \tag{3}$$

Traditional error bounds for trees involve a penalty proportional to $\sqrt{|T| \log n / n}$, where $|T|$ denotes the number of leaves in $T$ (see [12] or the "naive" bound in [2]). The penalty in (2) assigns a *different* weight to each leaf of the tree depending on both the depth of the leaf and the fraction of data reaching the leaf. Indeed, for very deep leaves, little data will reach those nodes, and such leaves will contribute very little to the overall penalty. For example, we may bound $\hat{p}'_A$ by $p'_A$ with high probability, and if $X$ has a bounded density, then $p'_A$ decays like $\max\{2^{-j}, \log n / n\}$, where $j$ is the depth of $A$. Thus, as $j$ increases, $[\![A]\!]$ grows additively with $j$, but $\hat{p}'_A$ decays at a multiplicative rate. The upshot is that the penalty $\Phi_n(T)$ *favors unbalanced trees*. Intuitively, if two trees have the same size and empirical error, minimizing the penalized empirical risk with this new penalty will select the tree that is more unbalanced, whereas a traditional penalty based only on tree size would not distinguish the two. This has advantages for classification because unbalanced trees are what we expect when approximating a lower dimensional decision boundary.

The derivation of (2) comes from applying standard concentration inequalities for sums of Bernoulli trials (most notably the relative Chernoff bound) in a spatially decomposed manner. Spatial decomposition allows the introduction of local probabilities $p_A$ to offset the complexity of each leaf node $A$. Our analysis is inspired by the work of Mansour and McAllester [2].

The uniform error bound of Theorem 1 can be converted (using standard techniques) into an oracle inequality that is the key to deriving rates of convergence for DDTs.

**Theorem 2.** *Let* $\widehat{T}_n$ *be as in (1) with* $\Phi_n$ *as in (2). Define*

$$\tilde{\Phi}_n(T) = \sum_{A \in \pi(T)} \sqrt{8 p'_A \frac{[\![A]\!] \log 2 + \log(2n)}{n}}.$$

*Then*

$$\mathbf{E}_{Z^n}\{R(\widehat{T}_n)\} - R^* \leq \min_{T \in \mathcal{T}} \left[ R(T) - R^* + 2\tilde{\Phi}_n(T) \right] + O\left(\frac{1}{n}\right). \tag{4}$$

Note that with high probability, $p'_A$ is an upper bound on $\hat{p}_A$, and therefore $\tilde{\Phi}_n$ upper bounds $\Phi_n$. The use of $\tilde{\Phi}_n$ instead of $\Phi_n$ in the oracle bound facilitates rate of convergence analysis. The oracle inequality essentially says that $\widehat{T}_n$ performs nearly as well as the DDT chosen by an oracle to minimize $R(T) - R^*$. The right hand side of (4) bears the interpretation of a decomposition into approximation error $(R(T) - R^*)$ and estimation error $\tilde{\Phi}_n(T)$.

## 4   Rates of Convergence

The classes of distributions we study are motivated by the work of Mammen and Tsybakov [4] and Tsybakov [1] which we now review. The classes are indexed by the smoothness

$\gamma$ of the Bayes decision boundary $\partial G^*$ and a parameter $\kappa$ that quantifies how "noisy" the distribution is near $\partial G^*$. We write $a_n \preccurlyeq b_n$ when $a_n = O(b_n)$ and $a_n \asymp b_n$ if both $a_n \preccurlyeq b_n$ and $b_n \preccurlyeq a_n$.

Let $\gamma > 0$, and take $r = \lceil \gamma \rceil - 1$ to be the largest integer not exceeding $\gamma$. Suppose $b : [0,1]^{d-1} \to [0,1]$ is $r$ times differentiable, and let $p_{b,s}$ denote the Taylor polynomial of $b$ of order $r$ at the point $s$. For a constant $c_1 > 0$, define $\Sigma(\gamma, c_1)$, the class of functions with Hölder regularity $\gamma$, to be the collection of all $b$ such that

$$|b(s') - p_{b,s}(s')| \le c_1 |s - s'|^\gamma \text{ for all } s, s' \in [0,1]^{d-1}.$$

Using Tsybakov's terminology, the Bayes decision set $G^*$ is a *boundary fragment* of smoothness $\gamma$ if $G^* = \text{epi}(b)$ for some $b \in \Sigma(\gamma, c_1)$. Here $\text{epi}(b) = \{(s,t) \in [0,1]^d : b(s) \le t\}$ is the epigraph of $b$. In other words, for a boundary fragment, the last coordinate of $\partial G^*$ is a Hölder-$\gamma$ function of the first $d-1$ coordinates.

Tsybakov also introduces a condition that characterizes the level of "noise" near $\partial G^*$ in terms of a noise exponent $\kappa$, $1 \le \kappa \le \infty$. Let $\Delta(f_1, f_2) = \{x \in [0,1]^d : f_1(x) \neq f_2(x)\}$. Let $c_2 > 0$. A distribution satisfies Tsybakov's noise condition with noise exponent $\kappa$ and constant $c_2$ if

$$\mathbf{P}_X(\Delta(f, f^*)) \le c_2(R(f) - R^*)^{1/\kappa} \quad \text{for all } f. \tag{5}$$

The case $\kappa = 1$ is the "low noise" case and corresponds to a jump of $\eta(x)$ at the Bayes decision boundary. The case $\kappa = \infty$ is the high noise case and imposes no constraint on the distribution (provided $c_2 \ge 1$). See [6] for further discussion.

Define the class $\mathcal{F} = \mathcal{F}(\gamma, \kappa) = \mathcal{F}(\gamma, \kappa, c_0, c_1, c_2)$ to be the collection of distributions of $Z = (X, Y)$ such that

**0A** For all measurable $A \subseteq [0,1]^d$, $\mathbf{P}_X(A) \le c_0 \lambda(A)$

**1A** $G^*$ is a boundary fragment defined by $b \in \Sigma(\gamma, c_1)$.

**2A** The margin condition is satisfied with noise exponent $\kappa$ and constant $c_2$.

Introducing the parameter $\rho = (d-1)/\gamma$, Tsybakov [1] proved the lower bound

$$\inf_{\widehat{f}_n} \sup_{\mathcal{F}} \left[ \mathbf{E}_{Z^n} \{ R(\widehat{f}_n) \} - R^* \right] \succcurlyeq n^{-\kappa/(2\kappa + \rho - 1)}. \tag{6}$$

The inf is over all rules for constructing classifiers from training data. Theoretical rules that achieve this lower bound are studied by [1, 4, 5, 13]. Unfortunately, none of these works provide computationally efficient algorithms for implementing the proposed discrimination rules, and it is unlikely that practical algorithms exist for these rules.

It is important to note that the lower bound in (6) is tight only when $\rho < 1$. To see this, fix $\rho > 1$. From the definition of $\mathcal{F}(\gamma, \kappa)$ we have $\mathcal{F}(\gamma, 1) \subset \mathcal{F}(\gamma, \kappa)$ for any $\kappa > 1$. As $\kappa \to \infty$, the right-hand side of (6) *decreases*. Therefore, the minimax rate for $\mathcal{F}(\gamma, \kappa)$ can be no faster than $n^{-1/(1+\rho)}$, which is the lower bound for $\mathcal{F}(\gamma, 1)$.

In light of the above, Tsybakov's noise condition does not improve the learning situation when $\rho > 1$. To achieve rates faster than $n^{-1/(1+\rho)}$ when $\rho > 1$, clearly an alternate assumption must be made. If the right-hand side of (6) is any indication, then the distributions responsible for slower rates are those with small $\kappa$. Thus, it would seem that we need a noise assumption that excludes those "low noise" distributions with small $\kappa$ that cause slower rates when $\rho > 1$.

Since recursive dyadic partitions can well-approximate $G^*$ with smoothness $\gamma \le 1$, we are in the regime of $\rho \ge (d-1)/\gamma \ge 1$. As motivated above, faster rates in this situation require an assumption that excludes low noise levels. We propose such an assumption. Like

Tsybakov's noise condition, our assumption is also defined in terms of constants $\kappa \geq 1$ and $c_2 > 0$. Because of limited space we are unable to fully present the modified noise condition, and we simply write

**2B** Low noise levels are excluded as defined in [11].

Effectively, **2B** says that the inequality in (5) is reversed, not for all classifiers $f$, but only for those $f$ that are the best DDT approximations to $f^*$ for each DDT resolutions parameter $M$. Using techniques presented in [13], we show in [11] that lower bounds of the form in (6) are valid when **2A** is replaced by **2B**. According to the results in the next section, these lower bounds are tight to within a log factor for $\rho > 1$.

## 5 Adaptive Rates for Dyadic Decision Trees

All of our rate of convergence proofs use the oracle inequality in the same basic way. The objective is to find an "oracle tree" $T^* \in \mathcal{T}$ such that both $R(T^*) - R^*$ and $\tilde{\Phi}_n(T^*)$ decay at the desired rate. This tree is roughly constructed as follows. First form a "regular" dyadic partition (the exact construction will depend on the specific problem) into cells of sidelength $1/m = 2^{-K}$, for a certain $K \leq L$. Then "prune back" all cells that do not intersect $\partial G^*$. Both approximation and estimation error may now be bounded using the given assumptions and elementary bounding methods. For example, $R(T^*) - R^* \preccurlyeq (\mathbf{P}_Z(\Delta(T^*, f^*)))^\kappa$ (by **2B**) $\preccurlyeq (\lambda(\Delta(T^*, f^*)))^\kappa$ (by **0A**) $\preccurlyeq m^{-\kappa}$ (by **1A**). This example reveals how the noise exponent enters the picture to affect the approximation error. See [11] for complete proofs.

### 5.1 Noise Adaptive Classification

Dyadic decision trees, selected according to the penalized empirical risk criterion discussed earlier, adapt to the unknown noise level to achieve faster rates as stated in Theorem 3 below. For now we focus on distributions with $\gamma = 1$ ($\rho = d - 1$), i.e., Lipschitz decision boundaries (the case $\gamma \neq 1$ is discussed in Section 5.4), and arbitrary noise parameter $\kappa$. The optimal rate for this class is $n^{-\kappa/(2\kappa+d-2)}$ [11]. We will see that DDTs can adaptively learn at a rate of $(\log n/n)^{\kappa/(2\kappa+d-2)}$.

In an effort to be more general and practical, we replace the boundary fragment condition **1A** with a less restrictive assumption. Tysbakov and van de Geer [5] assume the Bayes decision set $G^*$ is a boundary fragment, meaning it is known a priori that (a) one coordinate of $\partial G^*$ is a function of the others, (b) that coordinate is known, and (c) class 1 corresponds to the region *above* $\partial G^*$. The following condition includes all piecewise Lipschitz decision boundaries, and allows $\partial G^*$ to have arbitrary orientation and $G^*$ to have multiple connected components. Let $\mathcal{P}_m$ denote the regular partition of $[0,1]^d$ into hypercubes of sidelength $1/m$ where $m$ is a dyadic integer (i.e., a power of 2). A distribution of $Z$ satisfies the *box-counting* assumption with constant $c_1 > 0$ if

**1B** For all dyadic integers $m$, $\partial G^*$ intersects at most $c_1 m^{d-1}$ of the $m^d$ cells in $\mathcal{P}_m$.

Condition **1A** ($\gamma = 1$) implies **1B**, (with a different $c_1$) so the minimax rate under **0A**, **1B**, and **2B** is no faster than $n^{-\kappa/(2\kappa+d-2)}$.

**Theorem 3.** *Let $M \asymp (n/\log n)$. Take $\widehat{T}_n$ as in (1) with $\Phi_n$ as in (2). Then*

$$\sup \left[ \mathbf{E}_{Z^n}\{R(\widehat{T}_n)\} - R^* \right] \preccurlyeq \left( \frac{\log n}{n} \right)^{\frac{\kappa}{2\kappa+d-2}}. \tag{7}$$

*where the sup is over all distributions such that* **0A**, **1B**, *and* **2B** *hold.*

The complexity regularized DDT is adaptive in the sense that the noise exponent $\kappa$ and constants $c_0, c_1, c_2$ need not be known. $\widehat{T}_n$ can always be constructed and in opportune circumstances the rate in (7) is achieved.

## 5.2 When the Data Lie on a Manifold

For certain problems it may happen that the feature vectors lie on a manifold in the ambient space $\mathcal{X}$. When this happens, dyadic decision trees automatically adapt to achieve faster rates of convergence. To recast assumptions **0A** and **1B** in terms of a data manifold[1], we again use box-counting ideas. Let $c_0, c_1 > 0$ and $1 \leq d' \leq d$. The boundedness and regularity assumptions for a $d'$ dimensional manifold are given by

  **0B** For all dyadic integers $m$ and all $A \in \mathcal{P}_m$, $\mathbf{P}_X(A) \leq c_0 m^{-d'}$.

  **1C** For all dyadic integers $m$, $\partial G^*$ passes through at most $c_1 m^{d'-1}$ of the $m^d$ hypercubes in $\mathcal{P}_m$.

The minimax rate under these assumptions is $n^{-1/d'}$. To see this, consider the mapping of features $X' = (X^1, \ldots, X^{d'}) \in [0,1]^{d'}$ to $X = (X^1, \ldots, X^{d'}, 1/2, \ldots, 1/2) \in [0,1]^d$. Then $X$ lives on a $d'$ dimensional manifold, and clearly there can be no classifier achieving a rate faster than $n^{-1/d'}$ uniformly over all such $X$, as this would lead to a classifier outperforming the minimax rate for $X'$. As the following theorem shows, DDTs can achieve this rate to within a log factor.

**Theorem 4.** *Let $M \asymp (n/\log n)$. Take $\widehat{T}_n$ as in (1) with $\Phi_n$ as in (2). Then*

$$\sup \left[ \mathbf{E}_{Z^n}\{R(\widehat{T}_n)\} - R^* \right] \preccurlyeq \left( \frac{\log n}{n} \right)^{\frac{1}{d'}}. \tag{8}$$

*where the sup is over all distributions such that **0B** and **1C** hold.*

Again, $\widehat{T}_n$ is adaptive in that it does not require knowledge $d', c_0$, or $c_1$.

## 5.3 Irrelevant Features

The "relevant" data dimension is the number of relevant features/attributes, meaning the number $d'' < d$ of features of $X$ that are not independent of $Y$. By an argument like that in the previous section, the minimax rate under this assumption (and **0A** and **1B**) can be seen to be $n^{-1/d''}$. Once again, DDTs can achieve this rate to within a log factor.

**Theorem 5.** *Let $M \asymp (n/\log n)$. Take $\widehat{T}_n$ as in (1) with $\Phi_n$ as in (2).*

$$\sup \left[ \mathbf{E}_{Z^n}\{R(\widehat{T}_n)\} - R^* \right] \preccurlyeq \left( \frac{\log n}{n} \right)^{\frac{1}{d''}}. \tag{9}$$

*where the sup is over all distributions with relevant data dimension $d''$ and such that **0A** and **1B** hold.*

As in the previous theorems, our learning rule is adaptive in the sense that it does not need to be told $d''$ or which $d''$ features are relevant.

### 5.4 Adapting to Bayes Decision Boundary Smoothness

Our results thus far apply to Tsybakov's class with $\gamma = 1$. In [10] we show that DDTs with polynomial classifiers decorating the leaves can achieve faster rates for $\gamma > 1$. Combined with the analysis here, these rates can approach $n^{-1}$ under appropriate noise assumptions. Unfortunately, the rates we obtain are suboptimal and the classifiers are not practical.

For $\gamma \leq 1$, free DDTs adaptively attain the minimax rate (within a log factor) of $n^{-\gamma/(\gamma+d-1)}$. Due to space limitations, this discussion is deferred to [11]. Finding practical classifiers that adapt to the optimal rate for $\gamma > 1$ is a current line of research.

## 6 Conclusion

Dyadic decision trees adapt to a variation of Tsybakov's noise condition, data manifold dimension and the number of relevant features to achieve minimax optimal rates of convergence (to within a log factor). DDTs are constructed by a computationally efficient penalized empirical risk minimization procedure based on a novel, spatially adaptive, data-dependent penalty. Although we consider each condition separately so as to simplify the discussion, the conditions can be combined to yield a rate of $(\log n/n)^{\kappa/(2\kappa+d^*-2)}$ where $d^*$ is the dimension of the manifold supporting the relevant features.

## Footnotes

[1] For simplicity, we eliminate the margin assumption here and in subsequent sections, although it could be easily incorporated to yield faster adaptive rates.

### References

[1] A. B. Tsybakov, "Optimal aggregation of classifiers in statistical learning," *Ann. Stat.*, vol. 32, no. 1, pp. 135–166, 2004.

[2] Y. Mansour and D. McAllester, "Generalization bounds for decision trees," in *Proceedings of the Thirteenth Annual Conference on Computational Learning Theory*, N. Cesa-Bianchi and S. Goldman, Eds., Palo Alto, CA, 2000, pp. 69–74.

[3] Y. Yang, "Minimax nonparametric classification–Part I: Rates of convergence," *IEEE Trans. Inform. Theory*, vol. 45, no. 7, pp. 2271–2284, 1999.

[4] E. Mammen and A. B. Tsybakov, "Smooth discrimination analysis," *Ann. Stat.*, vol. 27, pp. 1808–1829, 1999.

[5] A. B. Tsybakov and S. A. van de Geer, "Square root penalty: adaptation to the margin in classification and in edge estimation," 2004, preprint.

[6] P. Bartlett, M. Jordan, and J. McAuliffe, "Convexity, classification, and risk bounds," Department of Statistics, U.C. Berkeley, Tech. Rep. 638, 2003, to appear in *Journal of the American Statistical Association*.

[7] G. Blanchard, G. Lugosi, and N. Vayatis, "On the rate of convergence of regularized boosting classifiers," *J. Machine Learning Research*, vol. 4, pp. 861–894, 2003.

[8] J. C. Scovel and I. Steinwart, "Fast rates for support vector machines," Los Alamos National Laboratory, Tech. Rep. LA-UR 03-9117, 2004.

[9] G. Blanchard, C. Schäfer, and Y. Rozenholc, "Oracle bounds and exact algorithm for dyadic classification trees," in *Learning Theory: 17th Annual Conference on Learning Theory, COLT 2004*, J. Shawe-Taylor and Y. Singer, Eds. Heidelberg: Springer-Verlag, 2004, pp. 378–392.

[10] C. Scott and R. Nowak, "Near-minimax optimal classification with dyadic classification trees," in *Advances in Neural Information Processing Systems 16*, S. Thrun, L. Saul, and B. Schölkopf, Eds. Cambridge, MA: MIT Press, 2004.

[11] ——, "Minimax optimal classification with dyadic decision trees," Rice University, Tech. Rep. TREE0403, 2004. [Online]. Available: http://www.stat.rice.edu/~cscott

[12] A. Nobel, "Analysis of a complexity based pruning scheme for classification trees," *IEEE Trans. Inform. Theory*, vol. 48, no. 8, pp. 2362–2368, 2002.

[13] J.-Y. Audibert, "PAC-Bayesian statistical learning theory," Ph.D. dissertation, Université Paris 6, June 2004.
